# Spike-based learning rules and stabilization of persistent neural activity

**Xiaohui Xie and H. Sebastian Seung**
Dept. of Brain & Cog. Sci., MIT, Cambridge, MA 02139
{xhxie, seung}@mit.edu

## Abstract

We analyze the conditions under which synaptic learning rules based on action potential timing can be approximated by learning rules based on firing rates. In particular, we consider a form of plasticity in which synapses depress when a presynaptic spike is followed by a postsynaptic spike, and potentiate with the opposite temporal ordering. Such *differential anti-Hebbian plasticity* can be approximated under certain conditions by a learning rule that depends on the time derivative of the postsynaptic firing rate. Such a learning rule acts to stabilize persistent neural activity patterns in recurrent neural networks.

## 1 INTRODUCTION

Recent experiments have demonstrated types of synaptic plasticity that depend on the temporal ordering of presynaptic and postsynaptic spiking. At cortical[1] and hippocampal[2] synapses, long-term potentiation is induced by repeated pairing of a presynaptic spike and a succeeding postsynaptic spike, while long-term depression results when the order is reversed. The dependence of the change in synaptic strength on the difference $\Delta t = t_{post} - t_{pre}$ between postsynaptic and presynaptic spike times has been measured quantitatively. This *pairing function*, sketched in Figure

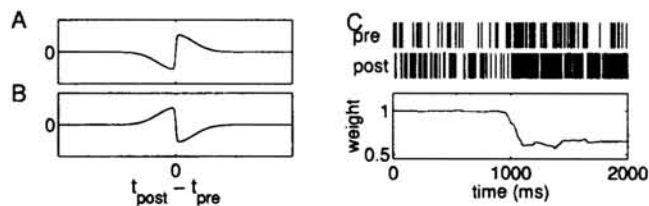

Figure 1: (A) Pairing function for differential Hebbian learning. The change in synaptic strength is plotted versus the time difference between postsynaptic and presynaptic spikes. (B) Pairing function for differential anti-Hebbian learning. (C) Differential anti-Hebbian learning is driven by changes in firing rates. The synaptic learning rule of Eq. (1) is applied to two Poisson spike trains. The synaptic strength remains roughly constant in time, except when the postsynaptic rate changes.

1A, has positive and negative lobes correspond to potentiation and depression, and a width of tens of milliseconds. We will refer to synaptic plasticity associated with this pairing function as differential Hebbian plasticity—*Hebbian* because the conditions for

potentiation are as predicted by Hebb[3], and *differential* because it is driven by the difference between the opposing processes of potentiation and depression.

The pairing function of Figure 1A is not characteristic of all synapses. For example, an opposite temporal dependence has been observed at electrosensory lobe synapses of electric fish[4]. As shown in Figure 1B, these synapses depress when a presynaptic spike is followed by a postsynaptic one, and potentiate when the order is reversed. We will refer to this as differential anti-Hebbian plasticity.

According to these experiments, the maximum ranges of the differential Hebbian and anti-Hebbian pairing functions are roughly 20 and 40 ms, respectively. This is fairly short, and seems more compatible with descriptions of neural activity based on spike timing rather than instantaneous firing rates[5, 6]. In fact, we will show that there are some conditions under which spike-based learning rules can be approximated by rate-based learning rules. Other people have also studied the relationship between spike-based and rate-based learning rules[7, 8].

The pairing functions of Figures 1A and 1B lead to rate-based learning rules like those traditionally used in neural networks, except that they depend on temporal derivatives of firing rates as well as firing rates themselves. We will argue that the differential anti-Hebbian learning rule of Figure 1B could be a general mechanism for tuning the strength of positive feedback in networks that maintain a short-term memory of an analog variable in persistent neural activity. A number of recurrent network models have been proposed to explain memory-related neural activity in motor [9] and prefrontal[10] cortical areas, as well as the head direction system [11] and oculomotor integrator[12, 13, 14]. All of these models require precise tuning of synaptic strengths in order to maintain continuously variable levels of persistent activity. As a simple illustration of tuning by differential anti-Hebbian learning, a model of persistent activity maintained by an integrate-and-fire neuron with an excitatory autapse is studied.

## 2   SPIKE-BASED LEARNING RULE

Pairing functions like those of Figure 1 have been measured using repeated pairing of a single presynaptic spike with a single postsynaptic spike. Quantitative measurements of synaptic changes due to more complex patterns of spiking activity have not yet been done. We will assume a simple model in which the synaptic change due to arbitrary spike trains is the sum of contributions from all possible pairings of presynaptic with postsynaptic spikes. The model is unlikely to be an exact description of real synapses, but could turn out to be approximately valid.

We will write the spike train of the $i$th neuron as a series of Dirac delta functions, $s_i(t) = \sum_n \delta(t - T_i^n)$, where $T_i^n$ is the $n$th spike time of the $i$th neuron. The synaptic weight from neuron $j$ to $i$ at time $t$ is denoted by $W_{ij}(t)$. Then the change in synaptic weight induced by presynaptic spikes occurring in the time interval $[0, T]$ is modeled as

$$W_{ij}(T + \lambda) - W_{ij}(\lambda) = \int_0^T dt_j \int_{-\infty}^\infty dt_i \, f(t_i - t_j) s_i(t_i) \, s_j(t_j) \tag{1}$$

Each presynaptic spike is paired with all postsynaptic spikes produced before and after. For each pairing, the synaptic weight is changed by an amount depending on the pairing function $f$. The pairing function is assumed to be nonzero inside the interval $[-\tau, \tau]$, and zero outside. We will refer to $\tau$ as the *pairing range*.

According to our model, each presynaptic spike results in induction of plasticity only after a latency $\lambda$. Accordingly, the arguments $T + \lambda$ and $\lambda$ of $W_{ij}$ on the left hand side of the equation are shifted relative to the limits $T$ and $0$ of the integral on the right hand side. We

will assume that the latency $\lambda$ is greater than the pairing range $\tau$, so that $W_{ij}$ at any time is only influenced by presynaptic and postsynaptic spikes that happened before that time, and therefore the learning rule is causal.

## 3  RELATION TO RATE-BASED LEARNING RULES

The learning rule of Eq. (1) is driven by correlations between presynaptic and postsynaptic activities. This dependence can be made explicit by making the change of variables $u = t_i - t_j$ in Eq. (1), which yields

$$W_{ij}(T + \lambda) - W_{ij}(\lambda) = \int_{-\tau}^{\tau} du\, f(u) C_{ij}(u) \tag{2}$$

where we have defined the cross-correlation

$$C_{ij}(u) = \int_{0}^{T} dt\, s_i(t + u)\, s_j(t) \;. \tag{3}$$

and made use of the fact that $f$ vanishes outside the interval $[-\tau, \tau]$. Our immediate goal is to relate Eq. (2) to learning rules that are based on the cross-correlation between firing rates,

$$C_{ij}^{rate}(u) = \int_{0}^{T} dt\, \nu_i(t + u)\, \nu_j(t) \tag{4}$$

There are a number of ways of defining instantaneous firing rates. Sometimes they are computed by averaging over repeated presentations of a stimulus. In other situations, they are defined by temporal filtering of spike trains. The following discussion is general, and should apply to these and other definitions of firing rates.

The "rate correlation" is commonly subtracted from the total correlation to obtain the "spike correlation" $C_{ij}^{spike} = C_{ij} - C_{ij}^{rate}$. To derive a rate-based approximation to the learning rule (2), we rewrite it as

$$W_{ij}(T + \lambda) - W_{ij}(\lambda) = \int_{-\tau}^{\tau} du\, f(u) C_{ij}^{rate}(u) + \int_{-\tau}^{\tau} du\, f(u) C_{ij}^{spike}(u) \tag{5}$$

and simply neglect the second term. Shortly we will discuss the conditions under which this is a good approximation. But first we derive another form for the first term by applying the approximation $\nu_i(t + u) \approx \nu_i(t) + u\dot{\nu}_i(t)$ to obtain

$$\int_{-\tau}^{\tau} du\, f(u) C_{ij}^{rate}(u) \approx \int_{0}^{T} dt[\beta_0 \nu_i(t) + \beta_1 \dot{\nu}_i(t)]\nu_j(t) \tag{6}$$

where we define

$$\beta_0 = \int_{-\tau}^{\tau} du\, f(u) \qquad \beta_1 = \int_{-\tau}^{\tau} du\, u f(u) \tag{7}$$

This approximation is good when firing rates vary slowly compared to the pairing range $\tau$. The learning rule depends on the postsynaptic rate through $\beta_0 \nu_i + \beta_1 \dot{\nu}_i$. When the first term dominates the second, then the learning rule is the conventional one based on correlations between firing rates, and the sign of $\beta_0$ determines whether the rule is Hebbian or anti-Hebbian.

In the remainder of the paper, we will discuss the more novel case where $\beta_0 = 0$. This holds for the pairing functions shown in Figures 1A and 1B, which have positive and negative lobes with areas that exactly cancel in the definition of $\beta_0$. Then the dependence on

postsynaptic activity is purely on the time derivative of the firing rate. Differential Hebbian learning corresponds to $\beta_1 > 0$ (Figure 1A), while differential anti-Hebbian learning leads to $\beta_1 < 0$ (Figure 1B). To summarize the $\beta_0 = 0$ case, the synaptic changes due to rate correlations are approximated by

$$\dot{W}_{ij} \propto \dot{\nu}_i \nu_j \quad \text{(diff. Hebbian)} \qquad \dot{W}_{ij} \propto -\dot{\nu}_i \nu_j \quad \text{(diff. anti-Hebbian)} \qquad (8)$$

for slowly varying rates. These formulas imply that a constant postsynaptic firing rate causes no net change in synaptic strength. Instead, changes in rate are required to induce synaptic plasticity.

To illustrate this point, Figure 1C shows the result of applying differential anti-Hebbian learning to two spike trains. The presynaptic spike train was generated by a 50 Hz Poisson process, while the postsynaptic spike train was generated by an inhomogeneous Poisson process with rate that shifted from 50 Hz to 200 Hz at 1 sec. Before and after the shift, the synaptic strength fluctuates but remains roughly constant. But the upward shift in firing rate causes a downward shift in synaptic strength, in accord with the sign of the differential anti-Hebbian rule in Eq. (8).

The rate-based approximation works well for this example, because the second term of Eq. (5) is not so important. Let us return to the issue of the general conditions under which this term can be neglected. With Poisson spike trains, the spike correlations $C_{ij}^{spike}(u)$ are zero in the limit $T \to \infty$, but for finite $T$ they fluctuate about zero. The integral over $u$ in the second term of (5) dampens these fluctuations. The amount of dampening depends on the pairing range $\tau$, which sets the limits of integration. In Figure 1C we used a relatively long pairing range of 100 ms, which made the fluctuations small even for small $T$. On the other hand, if $\tau$ were short, the fluctuations would be small only for large $T$. Averaging over large $T$ is relevant when the amplitude of $f$ is small, so that the rate of learning is slow. In this case, it takes a long time for significant synaptic changes to accumulate, so that plasticity is effectively driven by integrating over long time periods $T$ in Eq. (1).

In the brain, nonvanishing spike correlations are sometimes observed even in the $T \to \infty$ limit, unlike with Poisson spike trains. These correlations are often roughly symmetric about zero, in which case they should produce little plasticity if the pairing functions are antisymmetric as in Figures 1A and 1B. On the other hand, if the spike correlations are asymmetric, they could lead to substantial effects[6].

## 4   EFFECTS ON RECURRENT NETWORK DYNAMICS

The learning rules of Eq. (8) depend on both presynaptic and postsynaptic rates, like learning rules conventionally used in neural networks. They have the special feature that they depend on time derivatives, which has computational consequences for recurrent neural networks of the form

$$\dot{x}_i + x_i = \sum_j W_{ij} \sigma(x_j) + b_i \qquad (9)$$

Such classical neural network equations can be derived from more biophysically realistic models using the method of averaging[15] or a mean field approximation[16]. The firing rate of neuron $j$ is conventionally identified with $\nu_j = \sigma(x_j)$.

The cost function $E(\{x_i\}; \{W_{ij}\}) = \frac{1}{2} \sum_i \dot{\nu}_i^2$ quantifies the amount of drift in firing rate at the point $x_1, \ldots, x_N$ in the state space of the network. If we consider $\dot{\nu}_i$ to be a function of $x_i$ and $W_{ij}$ defined by (9), then the gradient of the cost function with respect to $W_{ij}$ is given by $\partial E / \partial W_{ij} = \sigma'(x_i) \dot{\nu}_i \nu_j$. Assuming that $\sigma$ is a monotonically increasing function so that $\sigma'(x_i) > 0$, it follows that the differential Hebbian update of (8) increases the cost function,

and hence increases the magnitude of the drift velocity. In contrast, the differential anti-Hebbian update decreases the drift velocity. This suggests that the differential anti-Hebbian update could be useful for creating fixed points of the network dynamics (9).

## 5 PERSISTENT ACTIVITY IN A SPIKING AUTAPSE MODEL

The preceding arguments about drift velocity were based on approximate rate-based descriptions of learning and network dynamics. It is important to implement spike-based learning in a spiking network dynamics, to check that our approximations are valid.

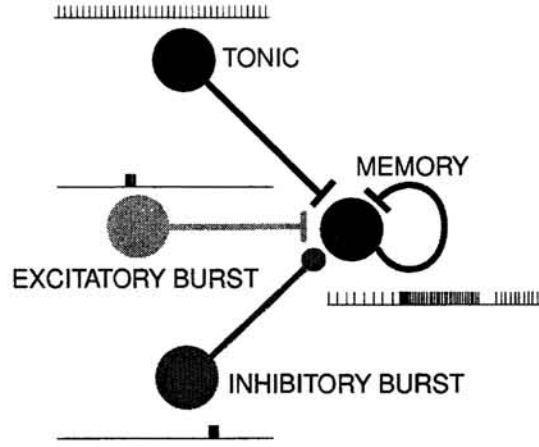

Therefore we have numerically simulated the simple recurrent circuit of integrate-and-fire neurons shown in Figure 2. The core of the circuit is the "memory neuron," which makes an excitatory autapse onto itself. It also receives synaptic input from three input neurons: a tonic neuron, an excitatory burst neuron, and an inhibitory burst neuron. It is known that this circuit can store a short-term memory of an analog variable in persistent activity, if the strengths of the autapse and tonic synapse are precisely tuned[17]. Here we show that this tuning can be accomplished by the spike-based learning rule of Eq. (1), with a differential anti-Hebbian pairing function like that of Figure 1B.

Figure 2: Circuit diagram for autapse model

The memory neuron is described by the equations

$$C_m \frac{dV}{dt} = -g_L(V - V_L) - g_E(V - V_E) - g_I(V - V_I) \tag{10}$$

$$\tau_{syn} \frac{dr}{dt} + r = \alpha_r \sum_n \delta(t - T_n) \tag{11}$$

where $V$ is the membrane potential. When $V$ reaches $V_{thres}$, a spike is considered to have occurred, and $V$ is reset to $V_{reset}$. Each spike at time $T_n$ causes a jump in the synaptic activation $r$ of size $\alpha_r/\tau_{syn}$, after which $r$ decays exponentially with time constant $\tau_{syn}$ until the next spike.

The synaptic conductances of the memory neuron are given by

$$g_E = Wr + W_0 r_0 + W_+ r_+ \qquad g_I = W_- r_- \tag{12}$$

The term $Wr$ is recurrent excitation from the autapse, where $W$ is the strength of the autapse. The synaptic activations $r_0$, $r_+$, and $r_-$ of the tonic, excitatory burst, and inhibitory burst neurons are governed by equations like (10) and (11), with a few differences. These neurons have no synaptic input; their firing patterns are instead determined by applied currents $I_{app,0}$, $I_{app,+}$ and $I_{app,-}$. The tonic neuron has a constant applied current, which makes it fire repetitively at roughly 20 Hz (Figure 3). For the excitatory and inhibitory burst neurons the applied current is normally zero, except for brief 100 ms current pulses that cause bursts of action potentials.

As shown in Figure 3, if the synaptic strengths $W$ and $W_0$ are arbitrarily set before learning, the burst neurons cause only transient changes in the firing rate of the memory neuron. After applying the spike-based learning rule (1) to tune both $W$ and $W_0$, the memory

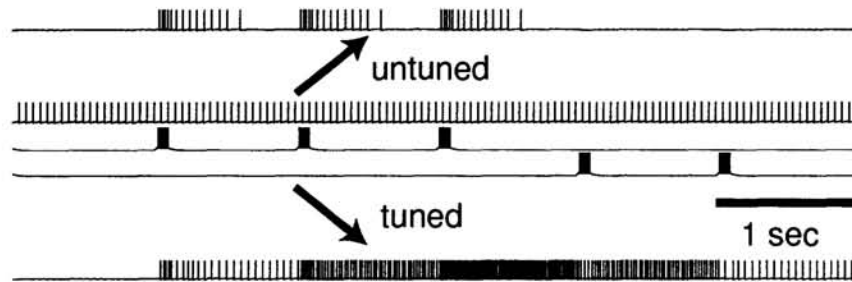

Figure 3: Untuned and tuned autapse activity. The middle three traces are the membrane potentials of the three input neurons in Figure 2 (spikes are drawn at the reset times of the integrate-and-fire neurons). Before learning, the activity of the memory neuron is not persistent, as shown in the top trace. After the spike-based learning rule (1) is applied to the synaptic weights $W$ and $W_0$, then the burst inputs cause persistent changes in activity. $C_m = 1$ nF, $g_L = 0.025$ $\mu$S, $V_L = -70$ mV, $V_E = 0$ mV, $V_I = -70$ mV, $V_{thres} = -52$ mV, $V_{reset} = -59$ mV, $\alpha_s = 1$, $\tau_{syn} = 100$ ms, $I_{app,0} = 0.5203$ nA, $I_{app,\pm} = 0$ or $0.95$ nA, $\tau_{syn,0} = 100$ ms, $\tau_{syn,+} = \tau_{syn,-} = 5$ ms, $W_+ = 0.1$, $W_- = 0.05$.

neuron is able to maintain persistent activity. During the interburst intervals (from $\lambda$ after one burst until $\lambda$ before the next), we made synaptic changes using the differential anti-Hebbian pairing function $f(t) = -A\sin(\pi t/\tau)$ for spike time differences in the range $[-\tau, \tau]$ with $A = 1.5 \times 10^{-4}$ and $\tau=\lambda=120$ ms. The resulting increase in persistence time can be seen in Figure 4A, along with the values of the synaptic weights versus time.

To quantify the performance of the system at maintaining persistent activity, we determined the relationship between $d\nu/dt$ and $\nu$ using a long sequence of interburst intervals, where $\nu$ was defined as the reciprocal of the interspike interval. If $W$ and $W_0$ are fixed at optimally tuned values, there is still a residual drift, as shown in Figure 4B. But if these parameters are allowed to adapt continuously, even after good tuning has been achieved, then the residual drift is even smaller in magnitude. This is because the learning rule tweaks the synaptic weights during each interburst interval, reducing the drift for that particular firing rate.

Autapse learning is driven by the autocorrelation of the spike train, rather than a cross-correlation. The peak in the autocorrelogram at zero lag has no effect, since the pairing function is zero at the origin. Since the autocorrelation is zero for small time lags, we used a fairly large pairing range in our simulations. In a recurrent network of many neurons, a shorter pairing range would suffice, as the cross-correlation does not vanish near zero.

## 6  DISCUSSION

We have shown that differential anti-Hebbian learning can tune a recurrent circuit to maintain persistent neural activity. This behavior can be understood by reducing the spike-based learning rule (1) to the rate-based learning rules of Eqs. (6) and (8). The rate-based approximations are good if two conditions are satisfied. First, the pairing range must be large, or the rate of learning must be slow. Second, spike synchrony must be weak, or have little effect on learning due to the shape of the pairing function.

The differential anti-Hebbian pairing function results in a learning rule that uses $-\dot{\nu}_i$ as a negative feedback signal to reduce the amount of drift in firing rate, as illustrated by our simulations of an integrate-and-fire neuron with an excitatory autapse. More generally, the learning rule could be relevant for tuning the strength of positive feedback in networks that maintain a short-term memory of an analog variable in persistent neural activity.

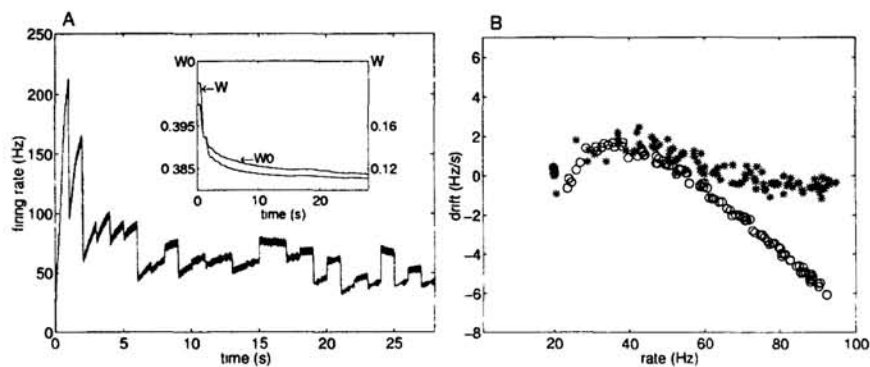

Figure 4: Tuning the autapse. (A) The persistence time of activity increases as the weights $W$ and $W_0$ are tuned. Each transition is driven by pseudorandom bursts of input (B) Systematic relationship between drift $d\nu/dt$ in firing rate and $\nu$, as measured from a long sequence of interburst intervals. If the weights are continuously fine-tuned ('*') the drift is less than with fixed well-tuned weights ('o').

For example, the learning rule could be used to improve the robustness of the oculomotor integrator[12, 13, 14] and head direction system[11] to mistuning of parameters. In deriving the differential forms of the learning rules in (8), we assumed that the areas under the positive and negative lobes of the pairing function are equal, so that the integral defining $\beta_0$ vanishes. In reality, this cancellation might not be exact. Then the ratio of $\beta_1$ and $\beta_0$ would limit the persistence time that can be achieved by the learning rule.

Both the oculomotor integrator and the head direction system are also able to integrate vestibular inputs to produce changes in activity patterns. The problem of finding generalizations of the present learning rules that train networks to integrate is still open.

# References

[1] H. Markram, J. Lubke, M. Frotscher, and B. Sakmann. *Science*, 275(5297):213–5, 1997.

[2] G. Q. Bi and M. M. Poo. *J Neurosci*, 18(24):10464–72, 1998.

[3] D. O. Hebb. *Organization of behavior*. Wiley, New York, 1949.

[4] C. C. Bell, V. Z. Han, Y. Sugawara, and K. Grant. *Nature*, 387(6630):278–81, 1997.

[5] W. Gerstner, R. Kempter, J. L. van Hemmen, and H. Wagner. *Nature*, 383(6595):76–81, 1996.

[6] L. F. Abbott and S. Song. *Adv. Neural Info. Proc. Syst.*, 11, 1999.

[7] P. D. Roberts. *J. Comput. Neurosci.*, 7:235-246, 1999.

[8] R. Kempter, W. Gerstner, and J. L. van Hemmen. *Phys. Rev. E*, 59(4):4498-4514, 1999.

[9] A. P. Georgopoulos, M. Taira, and A. Lukashin. *Science*, 260:47–52, 1993.

[10] M. Camperi and X. J. Wang. *J Comput Neurosci*, 5(4):383–405, 1998.

[11] K. Zhang. *J. Neurosci.*, 16:2112–2126, 1996.

[12] S. C. Cannon, D. A. Robinson, and S. Shamma. *Biol. Cybern.*, 49:127–136, 1983.

[13] H. S. Seung. *Proc. Natl. Acad. Sci. USA*, 93:13339–13344, 1996.

[14] H. S. Seung, D. D. Lee, B. Y. Reis, and D. W. Tank. *Neuron*, 2000.

[15] B. Ermentrout. *Neural Comput.*, 6:679–695, 1994.

[16] O. Shriki, D. Hansel, and H. Sompolinsky. *Soc. Neurosci. Abstr.*, 24:143, 1998.

[17] H. S. Seung, D. D. Lee, B. Y. Reis, and D. W. Tank. *J. Comput. Neurosci.*, 2000.
